# Edges are the 'Independent Components' of Natural Scenes.

Anthony J. Bell and Terrence J. Sejnowski
Computational Neurobiology Laboratory
The Salk Institute
10010 N. Torrey Pines Road
La Jolla, California 92037
tony@salk.edu, terry@salk.edu

## Abstract

Field (1994) has suggested that neurons with line and edge selectivities found in primary visual cortex of cats and monkeys form a sparse, distributed representation of natural scenes, and Barlow (1989) has reasoned that such responses should emerge from an unsupervised learning algorithm that attempts to find a factorial code of independent visual features. We show here that non-linear 'infomax', when applied to an ensemble of natural scenes, produces sets of visual filters that are localised and oriented. Some of these filters are Gabor-like and resemble those produced by the sparseness-maximisation network of Olshausen & Field (1996). In addition, the outputs of these filters are as independent as possible, since the infomax network is able to perform Independent Components Analysis (ICA). We compare the resulting ICA filters and their associated basis functions, with other decorrelating filters produced by Principal Components Analysis (PCA) and zero-phase whitening filters (ZCA). The ICA filters have more sparsely distributed (kurtotic) outputs on natural scenes. They also resemble the receptive fields of simple cells in visual cortex, which suggests that these neurons form an information-theoretic co-ordinate system for images.

## 1  Introduction.

Both the classic experiments of Hubel & Wiesel [8] on neurons in visual cortex, and several decades of theorising about feature detection in vision, have left open the question most succinctly phrased by Barlow "Why do we have edge detectors?" That is: are there any coding principles which would predict the formation of localised, oriented receptive

fields? Barlow's answer is that edges are suspicious coincidences in an image. Formalised information-theoretically, this means that our visual cortical feature detectors might be the end result of a *redundancy reduction* process [4, 2], in which the activation of each feature detector is supposed to be as *statistically independent* from the others as possible. Such a 'factorial code' potentially involves dependencies of all orders, but most studies [9, 10, 2] (and many others) have used only the second-order statistics required for *decorrelating* the outputs of a set of feature detectors. Yet there are multiple decorrelating solutions, including the 'global' unphysiological Fourier filters produced by PCA, so further constraints are required.

Field [7] has argued for the importance of sparse, or 'Minimum Entropy', coding [4], in which each feature detector is activated as rarely as possible. Olshausen & Field demonstrated [12] that such a sparseness criterion could be used to self-organise localised, oriented receptive fields.

Here we present similar results using a direct information-theoretic criterion which maximises the joint entropy of a non-linearly transformed output feature vector [5]. Under certain conditions, this process will perform Independent Component Analysis (or ICA) which is equivalent to Barlow's redundancy reduction problem. Since our ICA algorithm, applied to natural scenes, does produce local edge filters, Barlow's reasoning is vindicated. Our ICA filters are more sparsely distributed than those of other decorrelating filters, thus supporting some of the arguments of Field (1994) and helping to explain the results of Olshausen's network from an information-theoretic point of view.

## 2   Blind separation of natural images.

A perceptual system is exposed to a series of small image patches, drawn from an ensemble of larger images. In the *linear image synthesis* model [12], each image patch, represented by the vector $\mathbf{x}$, has been formed by the linear combination of $N$ basis functions. The basis functions form the columns of a fixed matrix, $\mathbf{A}$. The weighting of this linear combination (which varies with each image) is given by a vector, $\mathbf{s}$. Each component of this vector has its own associated basis function, and represents an underlying 'cause' of the image. Thus: $\mathbf{x=As}$. The goal of a perceptual system, in this simplified framework, is to linearly transform the images, $\mathbf{x}$, with a matrix of filters, $\mathbf{W}$, so that the resulting vector: $\mathbf{u= Wx}$, recovers the underlying causes, $\mathbf{s}$, possibly in a different order, and rescaled. For the sake of argument, we will define the ordering and scaling of the causes so that $\mathbf{W = A^{-1}}$. But what should determine their form? If we choose decorrelation, so that $\langle \mathbf{uu}^T \rangle = \mathbf{I}$, then the solution for $\mathbf{W}$ must satisfy:

$$\mathbf{W}^T\mathbf{W} = \langle \mathbf{xx}^T \rangle^{-1} \tag{1}$$

There are several ways to constrain the solution to this:

(1) Principal Components Analysis $\mathbf{W}_P$ (PCA), is the Orthogonal (global) solution $[\mathbf{WW}^T = \mathbf{I}]$. The PCA solution to Eq.(1) is $\mathbf{W}_P = \mathbf{D}^{-\frac{1}{2}}\mathbf{E}^T$, where $\mathbf{D}$ is the diagonal matrix of eigenvalues, and $\mathbf{E}$ is the matrix who's columns are the eigenvectors. The filters (rows of $\mathbf{W}_P$) are orthogonal, are thus the same as the PCA basis functions, and are typically *global* Fourier filters, ordered according to the amplitude spectrum of the image. Example PCA filters are shown in Fig.1a.

(2) Zero-phase Components Analysis $\mathbf{W}_Z$ (ZCA), is the Symmetrical (local) solution $[\mathbf{WW}^T = \mathbf{W}^2]$. The ZCA solution to Eq.(1) is $\mathbf{W}_Z = \langle \mathbf{xx}^T \rangle^{-1/2}$ (matrix square root). ZCA is the polar opposite of PCA. It produces *local* (centre-surround type) whitening fil-

ters, which are ordered according to the phase spectrum of the image. That is, each filter whitens a given pixel in the image, preserving the spatial arrangement of the image and flattening its frequency (amplitude) spectrum. $\mathbf{W}_Z$ is related to the transforms described by Atick & Redlich [3]. Example ZCA filters and basis functions are shown in Fig.1b.

(3) Independent Components Analysis $\mathbf{W}_I$ (ICA), is the Factorised (semi-local) solution $[f_{\mathbf{u}}(\mathbf{u}) = \prod_i f_{u_i}(u_i)]$. Please see [5] for full references. The 'infomax' solution we describe here is related to the approaches in [5, 1, 6].

As we will show, in Section 5, ICA on natural images produces decorrelating filters which are sensitive to both phase (locality) and spatial frequency information, just as in transforms involving oriented Gabor functions or wavelets. Example ICA filters are shown in Fig.1d and their corresponding basis functions are shown in Fig.1e.

# 3  An ICA algorithm.

It is important to recognise two differences between finding an ICA solution, $\mathbf{W}_I$, and other decorrelation methods. (1) there may be no ICA solution, and (2) a given ICA algorithm may not find the solution even if it exists, since there are approximations involved. In these senses, ICA is different from PCA and ZCA, and cannot be calculated analytically, for example, from second order statistics (the covariance matrix), except in the gaussian case.

The approach which we developed in [5] (see there for further references to ICA) was to maximise by stochastic gradient ascent the joint entropy, $H[g(\mathbf{u})]$, of the linear transform squashed by a sigmoidal function, $g$. When the non-linear function is the same (up to scaling and shifting) as the cumulative density functions (c.d.f.s) of the underlying independent components, it can be shown (Nadal & Parga 1995) that such a non-linear 'infomax' procedure also minimises the mutual information between the $u_i$, exactly what is required for ICA. In most cases, however, we must pick a non-linearity, $g$, without any detailed knowledge of the probability density functions (p.d.f.s) of the underlying independent components. In cases where the p.d.f.s are super-gaussian (meaning they are peakier and longer-tailed than a gaussian, having kurtosis greater than 0), we have repeatedly observed, using the logistic or tanh nonlinearities, that maximisation of $H[g(\mathbf{u})]$ still leads to ICA solutions, when they exist, as with our experiments on speech signal separation [5]. Although the infomax algorithm is described here as an ICA algorithm, a fuller understanding needs to be developed of under exactly what conditions it may fail to converge to an ICA solution.

The basic infomax algorithm changes weights according to the entropy gradient. Defining $y_i = g(u_i)$ to be the sigmoidally transformed output variables, the stochastic gradient learning rule is:

$$\Delta \mathbf{W} \propto \frac{\partial H(\mathbf{y})}{\partial \mathbf{W}} = E\left[\frac{\partial \ln |J|}{\partial \mathbf{W}}\right] = E[\mathbf{W}^{-T} + \hat{\mathbf{y}}\mathbf{x}^T] \tag{2}$$

In this, $E$ denotes expected value, $\mathbf{y} = [g(u_1) \ldots g(u_N)]^T$, and $|J|$ is the absolute value of the determinant of the Jacobian matrix: $J = \det[\partial y_i/\partial x_j]_{ij}$, and $\hat{\mathbf{y}} = [\hat{y}_1 \ldots \hat{y}_N]^T$, the elements of which depend on the nonlinearity according to: $\hat{y}_i = \partial/\partial y_i(\partial y_i/\partial u_i)$.

Amari, Cichocki & Yang [1] have proposed a modification of this rule which utilises the *natural* gradient rather than the *absolute* gradient of $H(\mathbf{y})$. The natural gradient exists for objective functions which are functions of matrices, as in this case, and is the same as the *relative* gradient concept developed by Cardoso & Laheld (1996). It amounts to multiplying

the absolute gradient by $\mathbf{W}^T\mathbf{W}$, giving, in our case, the following altered version of Eq.(2):

$$\Delta\mathbf{W} \propto \frac{\partial H(\mathbf{y})}{\partial\mathbf{W}}\mathbf{W}^T\mathbf{W} = (\mathbf{I} + \hat{\mathbf{y}}\mathbf{u}^T)\mathbf{W} \tag{3}$$

This rule has the twin advantages over Eq.(2) of avoiding the matrix inverse, and of converging several orders of magnitude more quickly, for data, $\mathbf{x}$, that is not prewhitened. The speedup is explained by the fact that convergence is no longer dependent on the conditioning of the underlying basis function matrix, $\mathbf{A}$. Writing Eq.(3) for one weight gives $\Delta w_{ij} \propto w_{ij} + \hat{y}_i \sum_k w_{kj} u_k$. This rule is 'almost local' requiring a backwards pass.

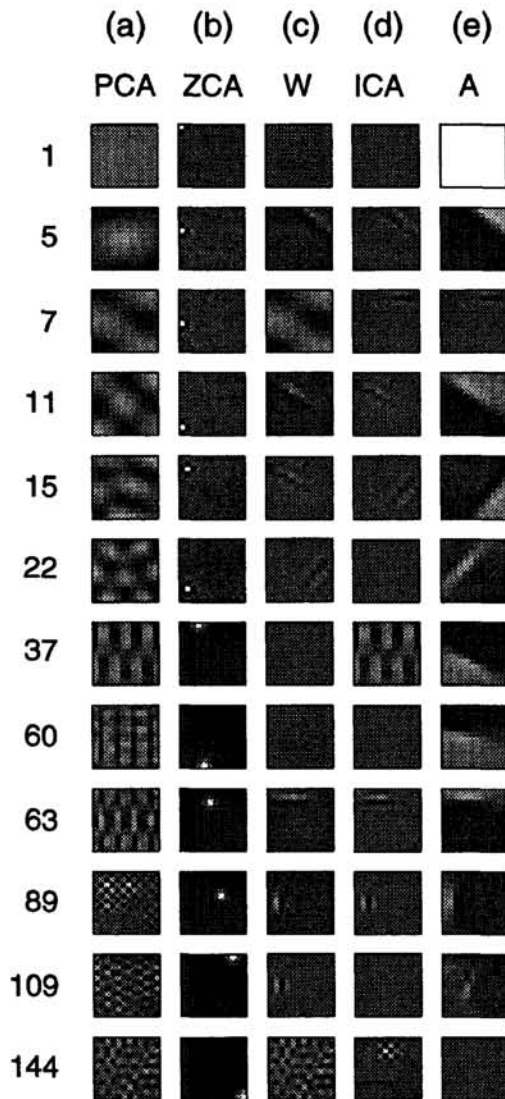

Figure 1: Selected decorrelating filters and their basis functions extracted from the natural scene data. Each type of decorrelating filter yielded 144 12x12 filters, of which we only display a subset here. Each column contains filters or basis functions of a particular type, and each of the rows has a number relating to which row of the filter or basis function matrix is displayed. (a) PCA ($\mathbf{W}_P$): The 1st, 5th, 7th etc Principal Components, showing increasing spatial frequency. There is no need to show basis functions and filters separately here since for PCA, they are the same thing. (b) ZCA ($\mathbf{W}_Z$): The first 6 entries in this column show the one-pixel wide centre-surround filter which whitens while preserving the phase spectrum. All are identical, but shifted. The lower 6 entries (37, 60 show the basis functions instead, which are the columns of the inverse of the $\mathbf{W}_Z$ matrix. (c) $\mathbf{W}$: the weights learnt by the ICA network trained on $\mathbf{W}_Z$-whitened data, showing (in descending order) the DC filter, localised oriented filters, and localised checkerboard filters. (d) $\mathbf{W}_I$: The corresponding ICA filters, calculated according to $\mathbf{W}_I = \mathbf{W}\mathbf{W}_Z$, looking like whitened versions of the $\mathbf{W}$-filters. (e) $\mathbf{A}$: the corresponding basis functions, or columns of $\mathbf{W}_I^{-1}$. These are the patterns which optimally stimulate their corresponding ICA filters, while not stimulating any other ICA filter, so that $\mathbf{W}_I\mathbf{A} = \mathbf{I}$.

## 4   Methods.

We took four natural scenes involving trees, leafs etc., and converted them to greyscale values between 0 and 255. A training set, $\{\mathbf{x}\}$, was then generated of 17,595 12x12 samples from the images. This was 'sphered' by subtracting the mean and multiplying by twice the local symmetrical (zero-phase) whitening filter: $\{\mathbf{x}\} \leftarrow 2\mathbf{W}_Z(\{\mathbf{x}\} - \langle\mathbf{x}\rangle)$. This removes both first and second order statistics from the data, and makes the covariance matrix of $\mathbf{x}$ equal to $4\mathbf{I}$. This is an appropriately scaled starting point for further training since infomax (Eq.(3)) on raw data, with the logistic function, $y_i = (1 + e^{-u_i})^{-1}$, produces a $\mathbf{u}$-vector which approximately satisfies $\langle\mathbf{uu}^T\rangle = 4\mathbf{I}$. Therefore, by prewhitening $\mathbf{x}$ in this way, we can ensure that the subsequent transformation, $\mathbf{u} = \mathbf{Wx}$, to be learnt should approximate an orthonormal matrix (rotation without scaling), roughly satisfying the relation $\mathbf{W}^T\mathbf{W} = \mathbf{I}$. The matrix, $\mathbf{W}$, is then initialised to the identity matrix, and trained using the logistic function version of Eq.(3), in which $\hat{y}_i = 1 - 2y_i$. Thirty sweeps through the data were performed, at the end of each of which, the order of the data vectors was permuted to avoid cyclical behaviour in the learning. In each sweep, the weights were updated in batches of 50 presentations. The learning rate (proportionality constant in Eq.(3)) followed 21 sweeps at 0.001, and 3 sweeps at each of 0.0005, 0.0002 and 0.0001, taking 2 hours running MATLAB on a Sparc-20 machine, though a reasonable result for 12x12 filters can be achieved in 30 minutes. To verify that the result was not affected by the starting condition of $\mathbf{W} = \mathbf{I}$, the training was repeated with several randomly initialised weight matrices, and also on data that was not prewhitened. The results were qualitatively similar, though convergence was much slower.

The full ICA transform from the raw image was calculated as the product of the sphering (ZCA) matrix and the learnt matrix: $\mathbf{W}_I = \mathbf{WW}_Z$. The basis function matrix, $\mathbf{A}$, was calculated as $\mathbf{W}_I^{-1}$. A PCA matrix, $\mathbf{W}_P$, was calculated. The original (unsphered) data was then transformed by all three decorrelating transforms, and for each the kurtosis of each of the 144 filters was calculated. Then the mean kurtosis for each filter type (ICA, PCA, ZCA) was calculated, averaging over all filters and input data.

## 5   Results.

The filters and basis functions resulting from training on natural scenes are displayed in Fig.1 and Fig.2. Fig.1 displays example filters and basis functions of each type. The PCA filters, Fig.1a, are spatially global and ordered in frequency. The ZCA filters and basis functions are spatially local and ordered in phase. The ICA filters, whether trained on the ZCA-whitened images, Fig.1c, or the original images, Fig.1d, are semi-local filters, most with a specific orientation preference. The basis functions, Fig.1e, calculated from the Fig.1d ICA filters, are not local, and naturally have the spatial frequency characteristics of the original images. Basis functions calculated from Fig.1d (as with PCA filters) are the same as the corresponding filters since the matrix $\mathbf{W}$ (as with $\mathbf{W}_P$) is orthogonal.

In order to show the full variety of ICA filters, Fig.2 shows, with lower resolution, all 144 filters in the matrix $\mathbf{W}$, in reverse order of the vector-lengths of the filters, so that the filters corresponding to higher-variance independent components appear at the top. The general result is that ICA filters are localised and mostly oriented. Unlike the basis functions displayed in Olshausen & Field (1996), they do not cover a broad range of spatial frequencies. However, the appropriate comparison to make is between the ICA basis functions, and the basis functions in Olshausen & Field's Figure 4. The ICA basis functions in our Fig.1e are

oriented, but not localised and therefore it is difficult to observe any multiscale properties. However, when we ran the ICA algorithm on Olshausen's images, which were preprocessed with a whitening/lowpass filter, our algorithm yielded basis functions which were localised multiscale Gabor patches qualitively similar to those in Olshausen's Figure 4. Part of the difference in our results is therefore attributable to different preprocessing techniques.

The distributions (image histograms) produced by PCA, ZCA and ICA are generally double-exponential ($e^{-|u_i|}$), or 'sparse', meaning peaky with a long tail, when compared to a gaussian, as predicted by Field [7]. The log histograms are seen to be roughly linear across 5 orders of magnitude. The histogram for the ICA filters, however, departs slightly from linearity, being more peaked, and having a longer tail than the ZCA and PCA histograms. This spreading of the tail signals the greater sparseness of the outputs of the ICA filters, and this is reflected in a calculated kurtosis measure of 10.04 for ICA, compared to 3.74 for PCA, and 4.5 for ZCA.

In conclusion, these simulations show that the filters found by the ICA algorithm of Eq.(3) with a logistic non-linearity are localised, oriented, and produce outputs distributions of very high sparseness. It is notable that this is achieved through an information theoretic learning rule which (1) has no noise model, (2) is sensitive to higher-order statistics (spatial coincidences), (3) is non-Hebbian (it is closer to anti-Hebbian) and (4) is simple enough to be almost locally implementable. Many other levels of higher-order invariance (translation, rotation, scaling, lighting) exist in natural scenes. It will be interesting to see if information-theoretic techniques can be extended to address these invariances.

## Acknowledgements

This work emerged through many extremely useful discussions with Bruno Olshausen and David Field. We are very grateful to them, and also to Paul Viola and Barak Pearlmutter. The work was supported by the Howard Hughes Medical Institute.

## References

[1] Amari S. Cichocki A. & Yang H.H. 1996. A new learning algorithm for blind signal separation, *Advances in Neural Information Processing Systems 8*, MIT press.

[2] Atick J.J. 1992. Could information theory provide an ecological theory of sensory processing? *Network* 3, 213-251

[3] Atick J.J. & Redlich A.N. 1993. Convergent algorithm for sensory receptive field development, *Neural Computation* 5, 45-60

[4] Barlow H.B. 1989. Unsupervised learning, *Neural Computation* 1, 295-311

[5] Bell A.J. & Sejnowski T.J. 1995. An information maximization approach to blind separation and blind deconvolution, *Neural Computation*, 7, 1129-1159

[6] Cardoso J-F. & Laheld B. 1996. Equivariant adaptive source separation, *IEEE Trans. on Signal Proc.*, Dec.1996.

[7] Field D.J. 1994. What is the goal of sensory coding? *Neural Computation* 6, 559-601

[8] Hubel D.H. & Wiesel T.N. 1968. Receptive fields and functional architecture of monkey striate cortex, *J. Physiol.*, 195: 215-244

[9] Linsker R. 1988. Self-organization in a perceptual network. *Computer*, 21, 105-117

[10] Miller K.D. 1988. Correlation-based models of neural development, in *Neuroscience and Connectionist Theory*, M. Gluck & D. Rumelhart, eds., 267-353, L.Erlbaum, NJ

[11] Nadal J-P. & Parga N. 1994. Non-linear neurons in the low noise limit: a factorial code maximises information transfer. *Network*, 5, 565-581.

[12] Olshausen B.A. & Field D.J. 1996. Natural image statistics and efficient coding, *Network: Computation in Neural Systems*, 7, 2.

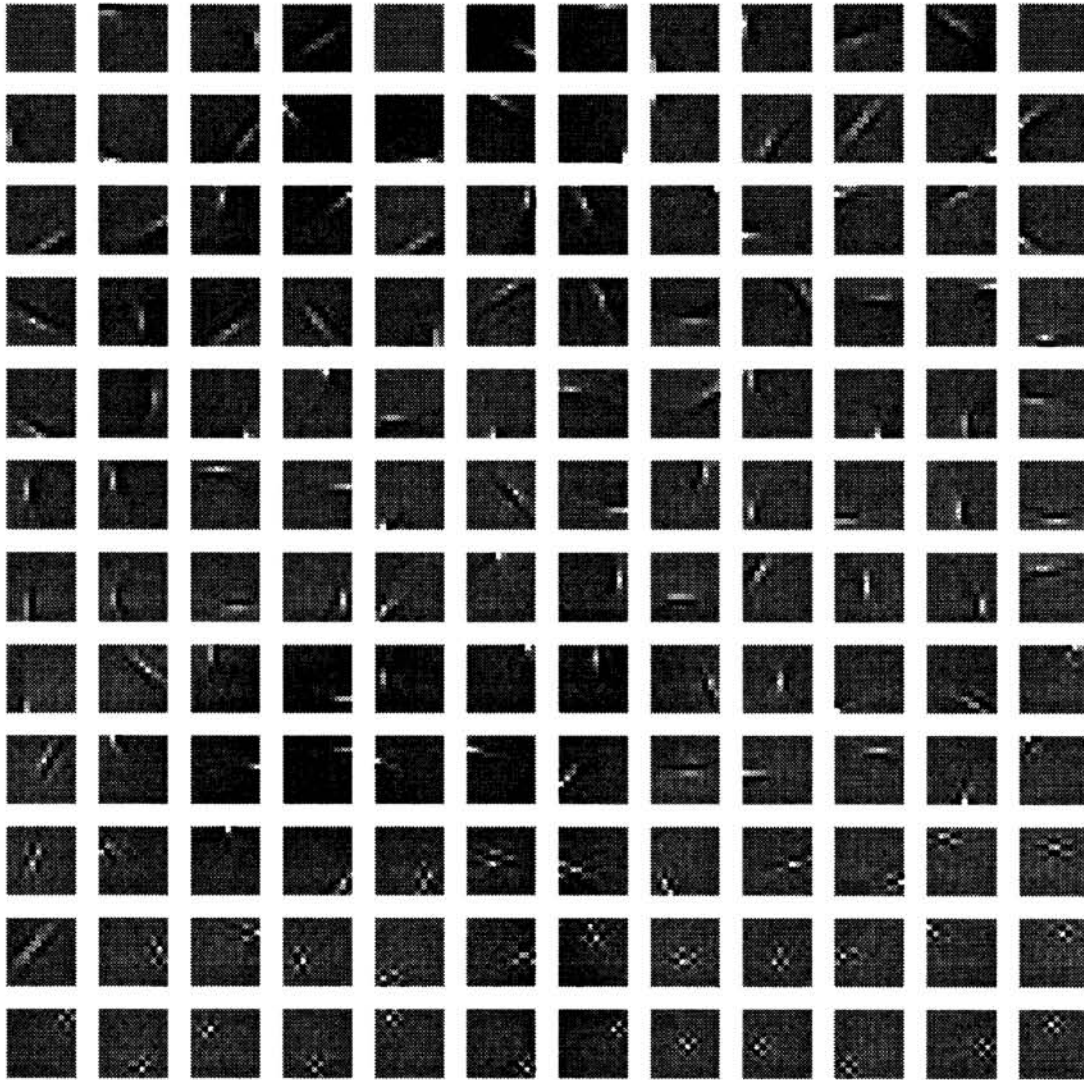

Figure 2: The matrix of 144 filters obtained by training on ZCA-whitened natural images. Each filter is a row of the matrix **W**, and they are ordered left-to-right, top-to-bottom in reverse order of the length of the filter vectors. In a rough characterisation, and more-or-less in order of appearance, the filters consist of one DC filter (top left), 106 oriented filters (of which 35 were diagonal, 37 were vertical and 34 horizontal), and 37 localised checkerboard patterns. The diagonal filters are longer than the vertical and horizontal due to the bias induced by having square, rather than circular, receptive fields.